# STATIC AND DYNAMIC ERROR PROPAGATION NETWORKS WITH APPLICATION TO SPEECH CODING

*A J Robinson, F Fallside*
Cambridge University Engineering Department
Trumpington Street, Cambridge, England

### Abstract

Error propagation nets have been shown to be able to learn a variety of tasks in which a static input pattern is mapped onto a static output pattern. This paper presents a generalisation of these nets to deal with time varying, or dynamic patterns, and three possible architectures are explored. As an example, dynamic nets are applied to the problem of speech coding, in which a time sequence of speech data are coded by one net and decoded by another. The use of dynamic nets gives a better signal to noise ratio than that achieved using static nets.

## 1. INTRODUCTION

This paper is based upon the use of the error propagation algorithm of Rumelhart, Hinton and Williams[1] to train a connectionist net. The net is defined as a set of units, each with an activation, and weights between units which determine the activations. The algorithm uses a gradient descent technique to calculate the direction by which each weight should be changed in order to minimise the summed squared difference between the desired output and the actual output. Using this algorithm it is believed that a net can be trained to make an arbitrary non-linear mapping of the input units onto the output units if given enough intermediate units. This 'static' net can be used as part of a larger system with more complex behaviour.

The static net has no memory for past inputs, but many problems require the context of the input in order to compute the answer. An extension to the static net is developed, the 'dynamic' net, which feeds back a section of the output to the input, so creating some internal storage for context, and allowing a far greater class of problems to be learned. Previously this method of training time dependence into nets has suffered from a computational requirement which increases linearly with the time span of the desired context. The three architectures for dynamic nets presented here overcome this difficulty.

To illustrate the power of these networks a general coder is developed and applied to the problem of speech coding. The non-linear solution found by training a dynamic net coder is compared with an established linear solution, and found to have an increased performance as measured by the signal to noise ratio.

## 2. STATIC ERROR PROPAGATION NETS

A static net is defined by a set of units and links between the units. Denoting $o_i$ as the value of the $i^{th}$ unit, and $w_{i,j}$ as the weight of the link between $o_i$ and $o_j$, we may divide up the units into input units, hidden units and output units. If we assign $o_0$ to a constant to form a

bias, the input units run from $o_1$ up to $o_{n_{inp}}$, followed by the hidden units to $o_{n_{hid}}$ and then the output units to $o_{n_{out}}$. The values of the input units are defined by the problem and the values of the remaining units are defined by:

$$\text{net}_i = \sum_{j=0}^{i-1} w_{i,j} o_j \qquad (2.1)$$

$$o_i = f(\text{net}_i) \qquad (2.2)$$

where $f(x)$ is any continuous monotonic non-linear function and is known as the activation function. The function used the application is:

$$f(x) = \frac{2}{1 + e^{-2x}} - 1 \qquad (2.3)$$

These equations define a net which has the maximum number of interconnections. This arrangement is commonly restricted to a layered structure in which units are only connected to the immediately preceding layer. The architecture of these nets is specified by the number of input, output and hidden units. Diagrammatically the static net is transformation of an input $u$, onto the output $y$, as in figure 1.

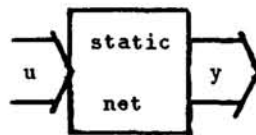

figure 1

The net is trained by using a gradient descent algorithm which minismises an energy term, $E$, defined as the summed squared error between the actual outputs, $o_i$, and the target outputs, $t_i$. The algorithm also defines an error signal, $\delta_i$, for each unit:

$$E = \frac{1}{2} \sum_{i=n_{hid}+1}^{n_{out}} (t_i - o_i)^2 \qquad (2.4)$$

$$\delta_i = f'(\text{net}_i)(t_i - o_i) \qquad n_{hid} < i \leq n_{out} \qquad (2.5)$$

$$= f'(\text{net}_i) \sum_{j=i+1}^{n_{out}} \delta_i w_{j,i} \qquad n_{inp} < i \leq n_{hid} \qquad (2.6)$$

where $f'(x)$ is the derivative of $f(x)$. The error signal and the activations of the units define the change in each weight, $\Delta w_{i,j}$.

$$\Delta w_{i,j} = \eta \delta_i o_j \qquad (2.7)$$

where $\eta$ is a constant of proportionality which determines the learning rate. The above equations define the error signal, $\delta_i$, for the input units as well as for the hidden units. Thus any number of static nets can be connected together, the values of $\delta_i$ being passed from input units of one net to output units of the preceding net. It is this ability of error propagation nets to be 'glued' together in this way that enables the construction of dynamic nets.

## 3. DYNAMIC ERROR PROPAGATION NETS

The essential quality of the dynamic net is is that its behaviour is determined both by the external input to the net, and also by its own internal state. This state is represented by the

activation of a group of units. These units form part of the output of a static net and also part of the input to another copy of the same static net in the next time period. Thus the state units link multiple copies of static nets over time to form a dynamic net.

## 3.1. DEVELOPMENT FROM LINEAR CONTROL THEORY

The analogy of a dynamic net in linear systems[2] may be stated as:

$$x_{p+1} = Ax_p + Bu_p \tag{3.1.1}$$
$$y_p = Cx_p \tag{3.1.2}$$

where $u_p$ is the input vector, $x_p$ the state vector, and $y_p$ the output vector at the integer time $p$. $A$, $B$ and $C$ are matrices.

The structure of the linear systems solution may be implemented as a non-linear dynamic net by substituting the matrices $A$, $B$ and $C$ by static nets, represented by the non-linear functions $A[.]$, $B[.]$ and $C[.]$. The summation operation of $Ax_p$ and $Bu_p$ could be achieved using a net with one node for each element in $x$ and $u$ and with unity weights from the two inputs to the identity activation function $f(x) = x$. Alternatively this net can be incorporated into the $A[.]$ net giving the architecture of figure 2.

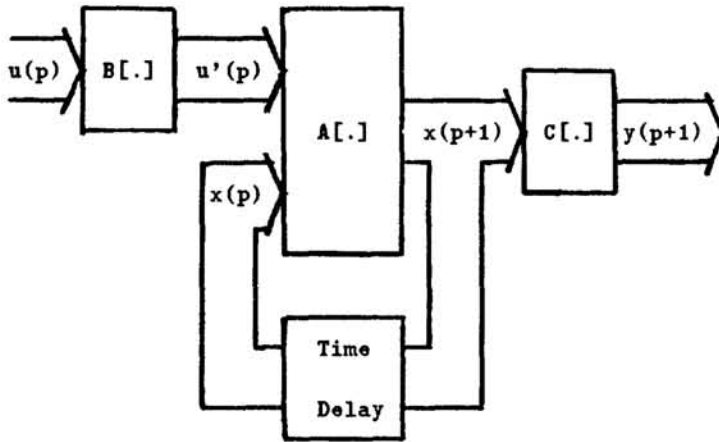
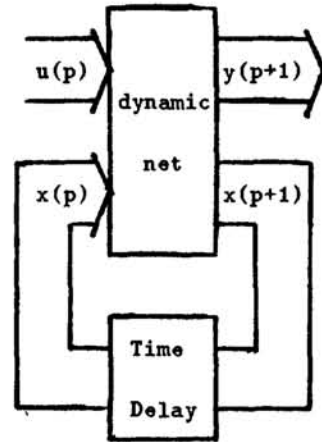

figure 2          figure 3

The three networks may be combined into one, as in figure 3. Simplicity of architecture is not just an aesthetic consideration. If three nets are used then each one must have enough computational power for its part of the task, combining the nets means that only the combined power must be sufficient and it allows common computations can be shared.

The error signal for the output $y_{p+1}$, can be calculated by comparison with the desired output. However, the error signal for the state units, $x_p$, is only given by the net at time $p+1$, which is not known at time $p$. Thus it is impossible to use a single backward pass to train this net. It is this difficulty which introduces the variation in the architectures of dynamic nets.

## 3.2. THE FINITE INPUT DURATION (FID) DYNAMIC NET

If the output of a dynamic net, $y_p$, is dependent on a finite number of previous inputs, $u_{p-P}$ to $u_p$, or if this assumption is a good approximation, then it is possible to formulate the

learning algorithm by expansion of the dynamic net for a finite time, as in figure 4. This formulation is simlar to a restricted version of the recurrent net of Rumelhart, Hinton and Williams.[1]

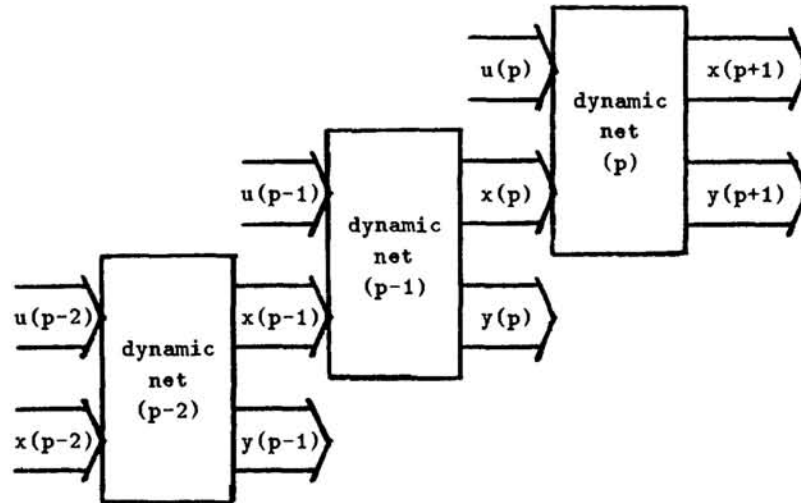

figure 4

Consider only the component of the error signal in past instantiations of the nets which is the result of the error signal at time $t$. The error signal for $y_p$ is calculated from the target output and the error signal for $x_p$ is zero. This combined error signal is propagated back though the dynamic net at $p$ to yield the error signals for $u_p$ and $x_p$. Similarly these error signals can then be propagated back through the net at $t-p$, and so on for all relevant inputs. The summed error signal is then used to change the weights as for a static net.

Formalising the FID dynamic net for a general time $q$, $q \leq p$:

$n_s$    is the number of state units
$o_{q,i}$    is the output value of unit $i$ at time $q$
$t_{q,i}$    is the target value of unit $i$ at time $q$
$\delta_{q,i}$    is the error value of unit $i$ at time $q$
$w_{i,j}$    is the weight between $o_i$ and $o_j$
$\Delta w_{q,i,j}$    is the weight change for this iteration at time $q$
$\Delta w_{i,j}$    is the total weight change for this iteration

These values are calculated in the same way as in a static net,

$$\text{net}_{q,i} = \sum_{j=0}^{i-1} w_{i,j} o_{q,j} \tag{3.2.1}$$

$$o_{q,i} = f(\text{net}_{q,i}) \tag{3.2.2}$$

$$\delta_{q,i} = f'(\text{net}_{q,i})(t_{q,i} - o_{q,i}) \qquad n_{\text{hid}} + n_s < i \leq n_{\text{out}} \tag{3.2.3}$$

$$= \delta_{q+1, i - n_{\text{hid}} + n_{\text{inp}} - n_s} \qquad n_{\text{hid}} < i \leq n_{\text{hid}} + n_s \tag{3.2.4}$$

$$= f'(\text{net}_{q,i}) \sum_{j=i+1}^{n_{\text{out}}} \delta_{q,j} w_{j,i} \qquad n_{\text{inp}} < i \leq n_{\text{hid}} \tag{3.2.5}$$

$$\Delta w_{q,i,j} = \eta \delta_{q,i} o_{q,j} \tag{3.2.6}$$

and the total weight change is given by the summation of the partial weight changes for all

previous times.

$$\Delta w_{i,j} \quad = \quad \sum_{q=p-P}^{p} \Delta w_{q,i,j} \tag{3.2.7}$$

$$= \quad \sum_{q=p-P}^{p} \eta \delta_{q,i} o_{q,j} \tag{3.2.8}$$

Thus, it is possible to train a dynamic net to incorporate the information from any time period of finite length, and so learn any function which has a finite impulse response.[*]

In some situations the approximation to a finite length may not be valid, or the storage and computational requirements of such a net may not be feasible. In such situations another approach is possible, the infinite input duration dynamic net.

### 3.3. THE INFINITE INPUT DURATION (IID) DYNAMIC NET

Although the forward pass of the FID net of the previous section is a non-linear process, the backward pass computes the effect of small variations on the forward pass, and is a linear process. Thus the recursive learning procedure described in the previous section may be compressed into a single operation.

Given the target values for the output of the net at time $p$, equations (3.2.3) and (3.2.4) define values of $\delta_{p,i}$ at the outputs. If we denote this set of $\delta_{p,i}$ by $D_p$ then equation (3.2.5) states that any $\delta_{p,i}$ in the net at time $p$ is simply a linear transformation of $D_p$. Writing the transformation matrix as $S$:

$$\delta_{p,i} \quad = \quad S_{p,i} D_p \tag{3.3.1}$$

In particular the set of $\delta_{p,i}$ which is to be fed back into the network at time $p-1$ is also a linear transformation of $D_p$

$$D_{p-1} \quad = \quad T_p D_p \tag{3.3.2}$$

or for an arbitrary time $q$:

$$D_q \quad = \quad \left( \prod_{r=q+1}^{p} T_r \right) D_p \tag{3.3.3}$$

so substituting equations (3.3.1) and (3.3.3) into equation (3.2.8):

$$\Delta w_{i,j} \quad = \quad \eta \sum_{q=-\infty}^{p} S_{q,i} \left( \prod_{r=q+1}^{p} T_r \right) D_p o_{q,j} \tag{3.3.4}$$

$$= \quad \eta M_{p,i,j} D_p \tag{3.3.5}$$

where:

$$M_{p,i,j} \quad = \quad \sum_{q=-\infty}^{p} S_{q,i} \left( \prod_{r=q+1}^{p} T_r \right) o_{q,j} \tag{3.3.6}$$

---

[*] This is a restriction on the class of functions which can be learned, the output will always be affected in some way by all previous inputs giving an infinite impulse response performance.

and note that $M_{p,i,j}$ can be written in terms of $M_{p-1,i,j}$ :

$$M_{p,i,j} = S_{p,i}\left(\prod_{r=p+1}^{t} T_r\right)o_{p,j} + \left(\sum_{q=-\infty}^{p-1} S_{q,i}\left(\prod_{r=q+1}^{p-1} T_r\right)o_{q,j}\right)T_p \quad (3.3.7)$$

$$= S_{p,i}o_{p,j} + M_{p-1,i,j}T_p \quad (3.3.8)$$

Hence we can calculate the weight changes for an infinite recursion using only the finite matrix $M$.

### 3.3. THE STATE COMPRESSION DYNAMIC NET

The previous architectures for dynamic nets rely on the propagation of the error signal back in time to define the format of the information in the state units. An alternative approach is to use another error propagation net to define the format of the state units. The overall architecture is given in figure 5.

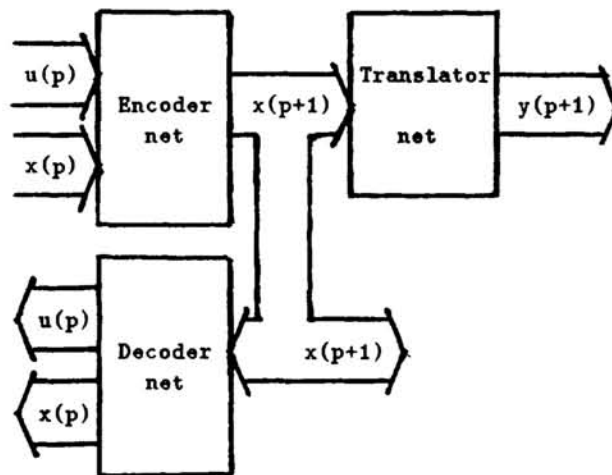

figure 5

The encoder net is trained to code the current input and current state onto the next state, while the decoder net is trained to do the reverse operation. The translator net codes the next state onto the desired output. This encoding/decoding attempts to represent the current input and the current state in the next state, and by the recursion, it will try to represent all previous inputs. Feeding errors back from the translator directs this coding of past inputs to those which are useful in forming the output.

### 3.4. COMPARISON OF DYNAMIC NET ARCHITECTURES

In comparing the three architectures for dynamic nets, it is important to consider the computational and memory requirements, and how these requirements scale with increasing context.

To train an FID net the net must store the past activations of the all the units within the time span of the necessary context. Using this minimal storage, the computational load scales proportionally to the time span considered, as for every new input/output pair the net must propagate an error signal back though all the past nets. However, if more sets of past activations are stored in a buffer, then it is possible to wait until this buffer is full before computing the weight changes. As the buffer size increases the computational load in

calculating the weight changes tends to that of a single backward pass through the units, and so becomes independent of the amount of context.

The largest matrix required to compute the IID net is $M$, which requires a factor of the number of outputs of the net more storage than the weight matrix. This must be updated on each iteration, a computational requirement larger than that of the FID net for small problems[3]. However, if this architecture were implemented on a parallel machine it would be possible to store the matrix $M$ in a distributed form over the processors, and locally calculate the weight changes. Thus, whilst the FID net requires the error signal to be propagated back in time in a strictly sequential manner, the IID net may be implemented in parallel, with possible advantages on parallel machines.

The state compression net has memory and computational requirements independent of the amount of context. This is achieved at the expense of storing recent information in the state units whether it is required to compute the output or not. This results in an increased computational and memory load over the more efficient FID net when implemented with a buffer for past outputs. However, the exclusion of external storage during training gives this architecture more biological plausibility, constrained of course by the plausibility of the error propagation algorithm itself.

With these considerations in mind, the FID net was chosen to investigate a 'real world' problem, that of the coding of the speech waveform.

# 4. APPLICATION TO SPEECH CODING

The problem of speech coding is one of finding a suitable model to remove redundancy and hence reduce the data rate of the speech. The Boltzmann machine learning algorithm has already been extended to deal to the dynamic case and applied to speech recognition[4]. However, previous use of error propagation nets for speech processing has mainly been restricted to explicit presentation of the context[5,6] or explicit feeding back the output units to the input[7,8], with some work done in using units with feedback links to themselves[9]. In a similar area, static error propagation nets have been used to perform image coding as well as conventional techniques[10].

## 4.1. THE ARCHITECTURE OF A GENERAL CODER

The coding principle used in this section is not restricted to coding speech data. The general problem is one of encoding the present input using past input context to form the transmitted signal, and decoding this signal using the context of the coded signals to regenerate the original input. Previous sections have shown that dynamic nets are able to represent context, so two dynamic nets in series form the architecture of the coder, as in figure 6.

This architecture may be specified by the number of input, state, hidden and transmission units. There are as many output units as input units and, in this application, both the transmitter and receiver have the same number of state and hidden units.

The input is combined with the internal state of the transmitter to form the coded signal, and then decoded by the receiver using its internal state. Training of the net involves the comparison of the input and output to form the error signal, which is then propagated back through past instantiations of the receiver and transmitter in the same way as a for a FID dynamic net.

It is useful to introduce noise into the coded signal during the training to reduce the information capacity of the transmission line. This forces the dynamic nets to incorporate time information, without this constraint both nets can learn a simple transformation without any time dependence. The noise can be used to simulate quantisation of the coded signal so

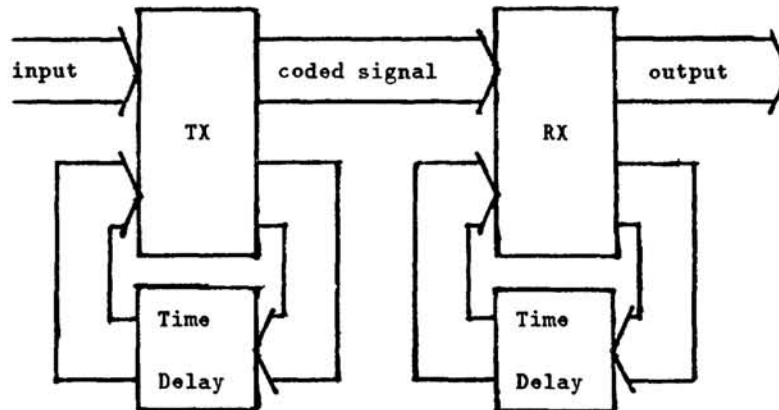

figure 6

quantifying the transmission rate. Unfortunately, a straight implementation of quantisation violates the requirement of the activation function to be continuous, which is necessary to train the net. Instead quantisation to $n$ levels may be simulated by adding a random value distributed uniformly in the range $+1/n$ to $-1/n$ to each of the channels in the coded signal.

## 4.2. TRAINING OF THE SPEECH CODER

The chosen problem was to present a single sample of digitised speech to the input, code to a single value quantised to fifteen levels, and then to reconstruct the original speech at the output. Fifteen levels was chosen as the point where there is a marked loss in the intelligibility of the speech, so implementation of these coding schemes gives an audible improvement. Two version of the coder net were implemented, both nets had eight hidden units, with no state units for the static time independent case and four state units for the dynamic time dependent case.

The data for this problem was 40 seconds of speech from a single male speaker, digitised to 12 bits at 10kHz and recorded in a laboratory environment. The speech was divided into two halves, the first was used for training and the second for testing.

The static and the dynamic versions of the architecture were trained on about 20 passes through the training data. After training the weights were frozen and the inclusion of random noise was replaced by true quantisation of the coded representation. A further pass was then made through the test data to yield the performance measurements.

The adaptive training algorithm of Chan[11] was used to dynamically alter the learning rates during training. Previously these machines were trained with fixed learning rates and weight update after every sample[3], and the use of the adaptive training algorithm has been found to result in a substantially deeper energy minima. Weights were updated after every 1000 samples, that is about 200 times in one pass of the training data.

## 4.3. COMPARISON OF PERFORMANCE

The performance of a coding schemes can be measured by defining the noise energy as half the summed squared difference between the actual output and the desired output. This energy is the quantity minimised by the error propagation algorithm. The lower the noise energy in relation to the energy of the signal, the higher the performance.

Three non-connectionist coding schemes were implemented for comparison with the static

and dynamic net coders. In the first the signal is linearly quantised within the dynamic range of the original signal. In the second the quantiser is restricted to operate over a reduced dynamic range, with values outside that range thresholded to the maximum and minimum outputs of the quantiser. The thresholds of the quantiser were chosen to optimise the signal to noise ratio. The third scheme used the technique of Differential Pulse Code Modulation (DPCM)[12] which involves a linear filter to predict the speech waveform, and the transmitted signal is the difference between the real signal and the predicted signal. Another linear filter reconstructs the original signal from the difference signal at the receiver. The filter order of the DPCM coder was chosen to be the same as the number of state units in the dynamic net coder, thus both coders can store the same amount of context enabling a comparison with this established technique.

The resulting noise energy when the signal energy was normalised to unity, and the corresponding signal to noise ratio are given in table 1 for the five coding techniques.

| coding method | normalised noise energy | signal to noise ratio in dB |
|---|---|---|
| linear, original thresholds | 0.071 | 11.5 |
| linear, optimum thresholds | 0.041 | 13.9 |
| static net | 0.049 | 13.1 |
| DPCM, optimum thresholds | 0.037 | 14.3 |
| dynamic net | 0.028 | 15.5 |

table 1

The static net may be compared with the two forms of the linear quantiser. Firstly note that a considerable improvement in the signal to noise ratio may be achieved by reducing the thresholds of the quantiser from the extremes of the input. This improvement is achieved because the distribution of samples in the input is concentrated around the mean value, with very few values near the extremes. Thus many samples are represented with greater accuracy at the expense of a few which are thresholded. The static net has a poorer performance than the linear quantiser with optimum thresholds. The form of the linear quantiser solution is within the class of problems which the static net can represent. It's failure to do so can be attributed to finding a local minima, a plateau in weight space, or corruption of the true steepest descent direction by noise introduced by updating the weights more than once per pass through the training data.

The dynamic net may be compared with the DPCM coding. The output from both these coders is no longer constrained to discrete signal levels and the resulting noise energy is lower than all the previous examples. The dynamic net has a significantly lower noise energy than any other coding scheme, although, from the static net example, this is unlikely to be an optimal solution. The dynamic net achieves a lower noise energy than the DPCM coder by virtue of the non-linear processing at each unit, and the flexibility of data storage in the state units.

As expected from the measured noise energies, there is an improvement in signal quality and intelligibility from the linear quantised speech through to the DCPM and dynamic net quantised speech.

## 5. CONCLUSION

This report has developed three architectures for dynamic nets. Each architecture can be formulated in a way where the computational requirement is independent of the degree of context necessary to learn the solution. The FID architecture appears most suitable for

implementation on a serial processor, the IID architecture has possible advantages for implementation on parallel processors, and the state compression net has a higher degree of biological plausibility.

Two FID dynamic nets have been coupled together to form a coder, and this has been applied to speech coding. Although the dynamic net coder is unlikely to have learned the optimum coding strategy, it does demonstrate that dynamic nets can be used to achieve an improved performance in a real world task over an established conventional technique.

One of the authors, A J Robinson, is supported by a maintenance grant from the U.K. Science and Engineering Research Council, and gratefully acknowledges this support.

# References

[1] D. E. Rumelhart, G. E. Hinton, and R. J. Williams. Learning internal representations by error propagation. In D. E. Rumelhart and J. L. McClelland, editors, *Parallel Distributed Processing: Explorations in the Microstructure of Cognition. Vol. 1: Foundations.*, Bradford Books/MIT Press, Cambridge, MA, 1986.

[2] O. L. R. Jacobs. *Introduction to Control Theory.* Clarendon Press, Oxford, 1974.

[3] A. J. Robinson and F. Fallside. *The Utility Driven Dynamic Error Propagation Network.* Technical Report CUED/F-INFENG/TR.1, Cambridge University Engineering Department, 1987.

[4] R. W. Prager, T. D. Harrison, and F. Fallside. Boltzmann machines for speech recognition. *Computer Speech and Language*, 1:3–27, 1986.

[5] J. L. Elman and D. Zipser. *Learning the Hidden Structure of Speech.* ICS Report 8701, University of California, San Diego, 1987.

[6] A. J. Robinson. *Speech Recognition with Associative Networks.* M.Phil Computer Speech and Language Processing thesis , Cambridge University Engineering Department, 1986.

[7] M. I. Jordan. *Serial Order: A Parallel Distributed Processing Approach.* ICS Report 8604, Institute for Cognitive Science, University of California, San Diego, May 1986.

[8] D. J. C. MacKay. *A Method of Increasing the Contextual Input to Adaptive Pattern Recognition Systems.* Technical Report RIPRREP/1000/14/87, Research Initiative in Pattern Recognition, RSRE, Malvern, 1987.

[9] R. L. Watrous, L. Shastri, and A. H. Waibel. Learned phonetic discrimination using connectionist networks. In J. Laver and M. A. Jack, editors, *Proceedings of the European Conference on Speech Technology*, CEP Consultants Ltd, Edinburgh, September 1987.

[10] G. W. Cottrell, P. Munro, and D Zipser. *Image Compression by Back Propagation: An Example of Existential Programming.* ICS Report 8702, Institute for Cognitive Science, University of California, San Diego, Febuary 1986.

[11] L. W. Chan and F. Fallside. *An Adaptive Learning Algorithm for Back Propagation Networks.* Technical Report CUED/F-INFENG/TR.2, Cambridge University Engineering Department, 1987, submitted to *Computer Speech and Language*.

[12] L. R. Rabiner and R. W. Schefer. *Digital Processing of Speech Signals.* Prentice Hall, Englewood Cliffs, New Jersey, 1978.
